# No evidence for active sparsification in the visual cortex

**Pietro Berkes, Benjamin L. White, and József Fiser**
Volen Center for Complex Systems
Brandeis University, Waltham, MA 02454

## Abstract

The proposal that cortical activity in the visual cortex is optimized for sparse neural activity is one of the most established ideas in computational neuroscience. However, direct experimental evidence for optimal sparse coding remains inconclusive, mostly due to the lack of reference values on which to judge the measured sparseness. Here we analyze neural responses to natural movies in the primary visual cortex of ferrets at different stages of development and of rats while awake and under different levels of anesthesia. In contrast with prediction from a sparse coding model, our data shows that population and lifetime sparseness decrease with visual experience, and increase from the awake to anesthetized state. These results suggest that the representation in the primary visual cortex is not actively optimized to maximize sparseness.

## 1 Introduction

It is widely believed that one of the main principles underlying functional organization of the early visual system is the reduction of the redundancy of relayed input from the retina. Such a transformation would form an optimally efficient code, in the sense that the amount of information transmitted to higher visual areas would be maximal. Sparse coding refers to a possible implementation of this general principle, whereby each stimulus is encoded by a small subset of neurons. This would allow the visual system to transmit information efficiently and with a small number of spikes, improving the signal-to-noise ratio, reducing the energy cost of encoding, improving the detection of "suspicious coincidences", and increasing storage capacity in associative memories [1, 2]. Computational models that optimize the sparseness of the responses of hidden units to natural images have been shown to reproduce the basic features of the receptive fields (RFs) of simple cells in V1 [3, 4, 5]. Moreover, manipulation of the statistics of the environment of developing animals leads to changes in the RF structure that can be predicted by sparse coding models [6].

Unfortunately, attempts to verify this principle experimentally have so far remained inconclusive. Electrophysiological studies performed in primary visual cortex agree in reporting high sparseness values for neural activity [7, 8, 9, 10, 11, 12]. However, it is contested whether the high degree of sparseness is due to a neural representation which is *optimally* sparse, or is an epiphenomenon due to neural selectivity [10, 12]. This controversy is mostly due to a lack of reference measurement with which to judge the sparseness of the neural representation in relative, rather than absolute terms. Another problem is that most of these studies have been performed on anesthetized animals [7, 9, 10, 11, 12], even though the effect of anesthesia might bias sparseness measurements (cf. Sec. 6).

In this paper, we report results from electrophysiological recordings from primary visual cortex (V1) of ferrets at various stages of development, from eye opening to adulthood, and of rats at different levels of anesthesia, from awake to deeply anesthetized, with the goal of testing the optimality of the neural code by studying changes in sparseness under different conditions. We compare this data

with theoretical predictions: 1) sparseness should *increase* with visual experience, and thus with age, as the visual system adapts to the statistics of the visual environment; 2) sparseness should be maximal in the "working regime" of the animal, i.e. for alert animals, and *decrease* with deeper levels of anesthesia. In both cases, the neural data shows a trend opposite to the one expected in a sparse coding system, suggesting that the visual system is not actively optimizing the sparseness of its representation.

The paper is organized as follows: We first introduce and discuss the lifetime and population sparseness measures we will be using throughout the paper. Next, we present the classical, linear sparse coding model of natural images, and derive an equivalent, stochastic neural network, whose output firing rates correspond to Monte Carlo samples from the posterior distribution of visual elements given an image. In the rest of the paper, we make use of this neural architecture in order to predict changes in sparseness over development and under anesthesia, and compare these predictions with electrophysiological recordings.

## 2  Lifetime and population sparseness

The diverse benefits of sparseness mentioned in the introduction rely on different aspects of the neural code, which are captured to a different extent by two sparseness measures, referred to as *lifetime* and *population sparseness*. *Lifetime sparseness* measures the distribution of the response of an individual cell to a set of stimuli, and is thus related to the cell's selectivity. This quantity characterizes the energy costs of coding with a set of neurons. On the other hand, the assessment of coding efficiency, as used by Treves and Rolls [13], is based upon the assumption that different stimuli activate small, distinct subsets of cells. These requirements of efficient coding are based upon the instantaneous population activity to stimuli and need to take into consideration the *population sparseness* of neural response. Average lifetime and population sparseness are identical if the units are statistically independent, in which case the distribution is called *ergodic* [10, 14]. In practice, neural dependencies (Fig. 3C) and residual dependencies in models [15] cause the two measures to be different.

Here we will use three measures of sparseness, two quantifying population sparseness, and one lifetime sparseness. To make a comparison with previous studies easier, we computed population and lifetime sparseness using a common measure introduced by Treves and Rolls [13] and perfected by Vinje and Gallant [8]:

$$\text{TR} = \left[ 1 - \frac{\left( \sum_{i=1}^{N} |r_i|/N \right)^2}{\sum_{i=1}^{N} r_i^2/N} \right] \bigg/ (1 - 1/N) \,, \tag{1}$$

where $r_i$ represents firing rates, and $i$ indexes time in the case of lifetime sparseness, and neurons for population sparseness. TR is defined between zero (less sparse) and one (more sparse), and depends on the shape of the distribution. For monotonic, non-negative distributions, such as that of firing rates, an exponential decay corresponds to $\text{TR} = 0.5$, and values smaller and larger than $0.5$ indicate distributions with lighter and heavier tails, respectively [14]. For population sparseness, we rescale the firing rate distribution by their standard deviation in time for the modelling results, and by $\sqrt{\sum_{t=1}^{T} r_t^2/T}$ for experimental data, as firing rate is non-negative. Moreover, in neural recordings we discard bins with no neural activity, as population TR is undefined in this case. TR does not depend on multiplicative changes firing rate, since it is invariant to rescaling the rates by a constant factor. However, it is not invariant to additive firing rate changes. This seems to be adequate for our purposes, as the arguments for sparseness involve metabolic costs and coding arguments like redundancy reduction that are sensitive to overall firing rates. Previous studies have shown that alternative measures of population and lifetime sparseness are highly correlated, therefore our choice does not affect the final results [15, 10].

We also report a second measure of population sparseness known as *activity sparseness* (AS), which is a direct translation of the definition of sparse codes as having a small number of neurons active at any time [15]:

$$\text{AS} = 1 - n_t/N \,, \tag{2}$$

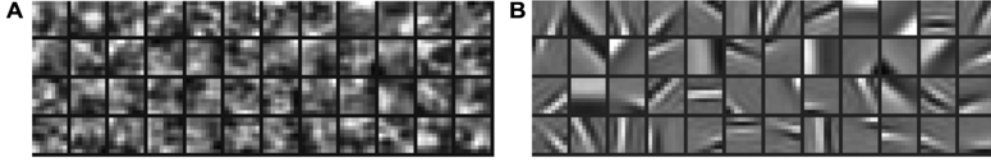

**Figure 1:** Generative weights of the sparse coding model at the beginning (A) and end (B) of learning.

where $n_t$ is defined as the number of neurons with activity larger than a given threshold at time $t$, and $N$ is the number of units. $AS = 1$ means that no neuron was active above the threshold, while $AS = 0$ means that all of all neurons were active. The threshold is set to be one standard deviation for the modeling results, or equivalently the upper 68th percentile of the distribution for neural firing rates. AS gives a very intuitive account of population sparseness, and is invariant to both multiplicative and additive changes in firing rate. However, since it discards most of the information about the shape of the distribution, it is a less sensitive measure than TR.

## 3 Sparse coding model

The sparseness assumption that natural scenes can be described by a small number of elements is generally translated in a model with sparsely distributed hidden units $x_k$, representing visual elements, that combine linearly to form an image $\mathbf{y}$ [3]:

$$p(x_k) = p_{sparse}(x_k) \propto \exp(f(x_k)), \qquad k = 1, \ldots, K \tag{3}$$

$$p(\mathbf{y}|\mathbf{x}) = \text{Normal}(\mathbf{y}; \mathbf{Gx}, \sigma_y^2), \tag{4}$$

where $K$ is the number of hidden units, $\mathbf{G}$ is the mixing matrix (also called the *generative weights*) and $\sigma_y^2$ is the variance of the input noise. Here we set the sparse prior distribution to a Student-t distribution with $\alpha$ degrees of freedom,

$$p(x_k) = \frac{1}{Z} \left( 1 + \frac{1}{\alpha} \left( \frac{x_k}{\lambda} \right)^2 \right)^{-\frac{\alpha+1}{2}}, \tag{5}$$

with $\lambda$ chosen such that the distribution has unit variance. This is a common prior for sparse coding models [3], and its analytical form allows the development of efficient inference and learning algorithms [16, 17].

The goal of learning is to adapt the model's parameters in order to best explain the observed data, i.e., to maximize the marginal likelihood

$$\sum_t \log p(\mathbf{y}_t|\mathbf{G}) = \sum_t \int \log p(\mathbf{y}_t|\mathbf{x}, \mathbf{G})p(\mathbf{x})\mathrm{d}\mathbf{x} \tag{6}$$

with respect to $\mathbf{G}$. We learn the weights using a Variational Expectation Maximization (VEM) algorithm, as described by Berkes *et al.* [17], with the difference that the generative weights are not treated as random variables, but as parameters with norm fixed to 1, in order to avoid potential confounds in successive analysis.

The model was applied to $9 \times 9$ pixel natural image patches, randomly chosen from 36 natural images from the van Hateren database, preprocessed as described in [5]. The dimensionality of the patches was reduced to 36 and the variances normalized by Principal Component Analysis. The model parameters were chosen to be $K = 48$ and $\alpha = 2.5$, a very sparse, slightly overcomplete representation. These parameters are very close to the ones that were found to be optimal for natural images [17]. The input noise was fixed to $\sigma_y^2 = 0.08$. The generative weights were initialized at random, with norm 1. We performed 1500 iterations of the VEM algorithm, using a new batch of 3600 patches at each iteration. Fig. 1 shows the generative weights at the start and at the end of learning. As expected from previous studies [3, 5], after learning the basis vectors are shaped like Gabor wavelets and resemble simple cell RFs.

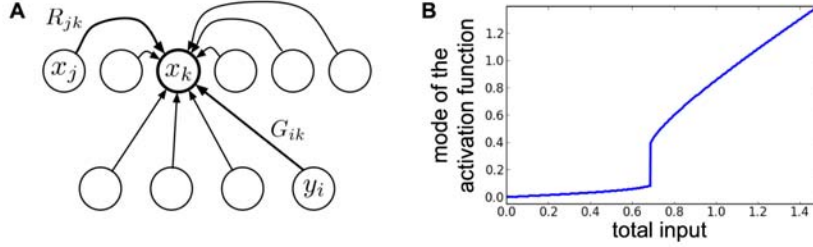

**Figure 2:** Neural implementation of Gibbs sampling in a sparse coding model. A) Neural network architecture. B) Mode of the activation probability of a neuron as a function of the total (feed-forward and recurrent) input, for a Student-t prior with $\alpha = 2.05$ and unit variance.

## 4   Sampling, sparse coding neural network

In order to gain some intuition about the neural operations that may underlie inference in this model, we derive an equivalent neural network architecture. It has been suggested that neural activity is best interpreted as samples from the posterior probability of an internal, probabilistic model of the sensory input. This assumption is consistent with many experimental observations, including high trial-by-trial variability and spontaneous activity in awake animals [18, 19, 20]. Moreover, sampling can be performed in parallel and asynchronously, making it suitable for a neural architecture. Assuming that neural activity corresponds to Gibbs sampling from the posterior probability over visual elements in the sparse coding model, we obtain the following expression for the distribution of the firing rate of a neuron, given a visual stimulus and the current state of the other neurons representing the image [18]:

$$p(x_k|x_{i \neq k}, \mathbf{y}) \propto p(\mathbf{y}|\mathbf{x})p(x_k) \tag{7}$$

$$\propto \exp\left(-\frac{1}{2\sigma_y^2}(\mathbf{y}^T\mathbf{y} - 2\mathbf{y}^T\mathbf{G}\mathbf{x} - \mathbf{x}^T\mathbf{R}\mathbf{x}) + f(x_k)\right) , \tag{8}$$

where $\mathbf{R} = -\mathbf{G}^T\mathbf{G}$. Expanding the exponent, eliminating the terms that do not depend on $x_k$, and noting that $R_{kk} = -1$, since the generative weights have unit norm, we get

$$p(x_k|x_{i \neq k}, \mathbf{y}) \propto \exp\left(\frac{1}{\sigma_y^2}(\sum_i G_{ik}y_i)x_k + \frac{1}{\sigma_y^2}(\sum_{j \neq k} R_{jk}x_j)x_k - \frac{1}{2\sigma_y^2}x_k^2 + f(x_k)\right) . \tag{9}$$

Sampling in a sparse coding model can thus be achieved by a simple neural network, where the $k$-th neuron integrates visual information through feed–forward connections from input $y_i$ with weights $G_{ik}/\sigma_y^2$, and information from other neurons via recurrent connections $R_{jk}/\sigma_y^2$ (Fig. 2A). Neural activity is then generated stochastically according to Eq. 9: The exponential activation function gives higher probability to higher rates with increasing input to the neuron, while the terms depending on $x_k^2$ and $f(x_k)$ penalize large firing rates. Fig. 2B shows the mode of the activation probability (Eq. 9) as a function of the total input to a neuron.

## 5   Active sparsification over learning

A simple, intuitive prediction for a system that optimizes for sparseness is that the sparseness of its representation should increase over learning. Since a sparse coding system, including our model, might not directly maximize our measures of sparseness, TR and AS, we verify this intuition by analyzing the model's representation of images at various stages of learning. We selected at random a new set of 1800 patches to be used as test stimuli. For every patch, we collected 50 Monte Carlo samples, using Gibbs sampling (Eq. 9) combined with an annealing scheme that starts by drawing samples from the model's prior distribution and continues to sample as the prior is deformed into the posterior [21]. This procedure ensures that the final samples come from the whole posterior distribution, which is highly multimodal in overcomplete models, and therefore that our analysis is not

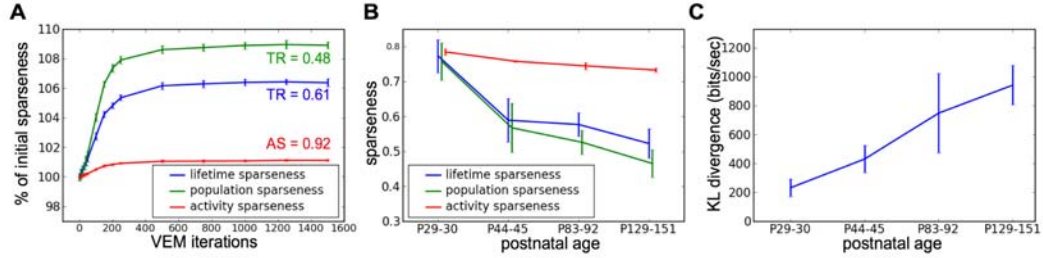

**Figure 3:** Development of sparseness, (A) over learning for the sparse coding model of natural images and (B) over age for neural responses in ferrets. (A) The lines indicate the average sparseness over units and samples. Error bars are one standard deviation over samples. Since the three measures have very different values, we report the change in sparseness in percent of the first iteration. Colored text: absolute values of sparseness at the end of learning. (B) The lines indicate the average sparseness for different animals. Error bars represent standard error of the mean (SEM). (C) KL divergence between the distribution of neural responses and the factorized distribution of neural responses. Error bars are SEM.

biased by the posterior distribution becoming more (or less) complex over learning. Fig. 3A shows the evolution of sparseness with learning. As anticipated, both population and lifetime sparseness increase monotonically.

Having confirmed our intuition with the sparse coding model, we turn to data from electrophysiological recordings. We analyzed multi-unit recordings from arrays of 16 electrodes implanted in the primary visual cortex of 15 ferrets at various stages of development, from eye opening at postnatal day 29 or 30 (P29-30) to adulthood at P151 (see Suppl Mat for experimental details). Over this maturation period, the visual system of ferrets adapts to the statistics of the environment [22, 23]. For each animal, neural activity was recorded and collected in 10 ms bins for 15 sessions of 100 seconds each (for a total of 25 minutes), during which the animal was shown scenes from a movie. We find that all three measures of sparseness decrease significantly with age[1]. Thus, during a period when the cortex actively adapts to the visual environment, the representation in primary visual cortex becomes *less* sparse, suggesting that the optimization of sparseness is not a primary objective for learning in the visual system. The decrease in population sparseness seems to be due to an increase in the dependencies between neurons: Fig. 3C shows the Kullback-Leibler divergence between the joint distribution $P$ of neural activity in 2 ms bins and the same distribution, factorized to eliminate neural dependencies, i.e., $\tilde{P}(r_1, \ldots r_N) := \prod_{i=1}^{N} P(r_i)$. The KL divergence increases with age (Spearman's $\rho = 0.73$, $P < 0.01$), indicating an increase in neural dependencies.

## 6 Active sparsification and anesthesia

The sparse coding neural network architecture of Fig. 2 makes explicit that an optimal sparse coding representation requires a process of *active sparsification*: In general, because of input noise and the overcompleteness of the representation, there are multiple possible combinations of visual elements that could account for a given image. To select among these combinations the most sparse solution, a competition between possible alternative interpretations must occur.

Consider for example a simple system with one input variable and two hidden units, such that $y = x_1 + 1.3 \cdot x_2 + \epsilon$, with Gaussian noise $\epsilon$. Given an observed value, $y$, there are infinitely many solutions to this equality, as shown by the dotted line in Fig. 4B for $y = 2$. These stimulus–induced correlations in the posterior are known as *explaining away*. Among all the solutions, the ones compatible with the sparse prior over $x_1$ and $x_2$ are given higher probability, giving raise to a bimodal

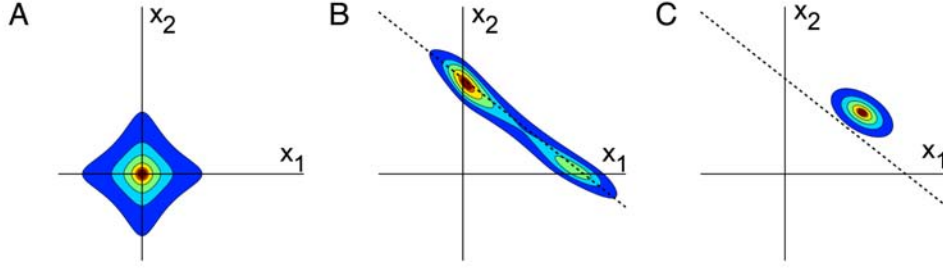

**Figure 4:** Active sparsification. Contour lines correspond to the 5, 25, 50, 75, 90, and 95 percentile of the distributions. A) Prior probability. B) Posterior probability given observed value $y = 2$. The dotted line indicates all solutions to $2 = x_1 + 1.3 \cdot x_2$. C) Posterior probability with weakened recurrent weights ($\alpha = 0.5$).

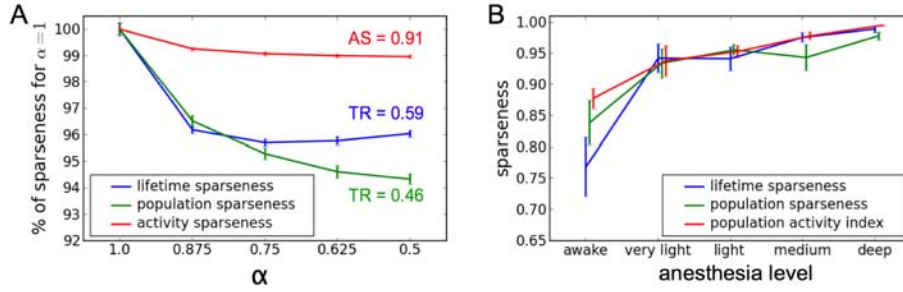

**Figure 5:** Active sparsification and anesthesia. A) Percent change in sparseness as the recurrent connections are weakened for various values of $\alpha$. Error bars are one standard deviation over samples. Colored text: absolute values of sparseness at the end of learning. B) Average sparseness measures for V1 responses at various levels of anesthesia. Error bars are SEM.

distribution centered around the two sparse solutions $x_1 = 0$, $x_2 = 1.54$, and $x_1 = 2$, $x_2 = 0$. From Eq. 9, it is clear that the recurrent connections are necessary in order to keep the activity of the neurons on the solution line, while the stochastic activation function makes sparse neural responses more likely. This active sparsification process is stronger for overcomplete representations, for when the generative weights are non-orthogonal (in which cases $|r_{ij}| \gg 0$), and for when the input noise is large, which makes the contribution from the prior more important.

In a system that optimizes sparseness, disrupting the active sparsification process will lead to lower lifetime and population sparseness. For example, if we reduce the strength of the recurrent connections in the neural network architecture (Eq. 9) by a factor $\alpha$,

$$p(x_k|x_{i \neq k}, \mathbf{y}) \propto \exp \left( \frac{1}{\sigma_y^2} (\sum_i G_{ik} y_i) x_k + \frac{1}{\sigma_y^2} \alpha (\sum_{j \neq k} R_{jk} x_j) x_k - \frac{1}{2\sigma_y^2} x_k^2 + f(x_k) \right) , \quad (10)$$

the neurons become more decoupled, and try to separately account for the input, as illustrated in Fig. 4C. The decoupling will result in a reduction of population sparseness, as multiple neurons become active to explain the same input. Also, lifetime sparseness will decrease, as the lack of competition between units means that individual units will be active more often.

Fig. 5 shows the effect of reducing the strength of recurrent connections in the model of natural images. We analyzed the parameters of the sparse coding model at the end of learning, and substituted the Gibbs sampling posterior distribution of Eq. 9 with the one in Eq. 10 for various values of $\alpha$. As predicted, decreasing $\alpha$ leads to a decrease in all sparseness measures.

We argue that a similar disruption of the active sparsification process can be obtained in electrophysiological experiments by comparing neural responses at different levels of isoflurane anesthesia. In general, the evoked, feed-forward responses of V1 neurons under anesthesia are thought to remain

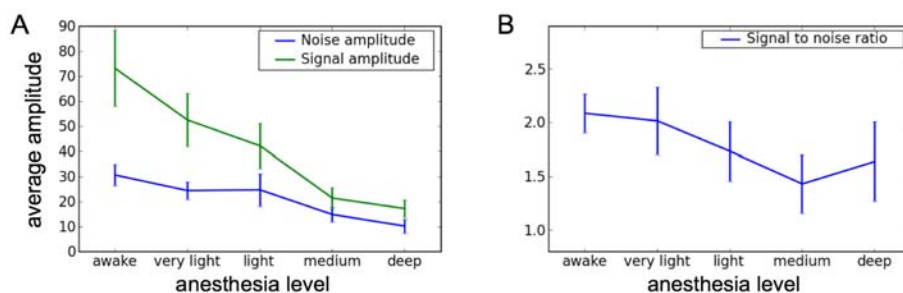

**Figure 6:** Neuronal response to a 3.75 Hz full-field stimulation under different levels of anesthesia. Error bars are SEM. A) Signal and noise amplitudes. B) Signal-to-noise ratio.

largely intact: Despite a decrease in average firing rate, the selectivity of neurons to orientation, frequency, and direction of motion has been shown to be very similar in awake and anesthetized animals [24, 25, 26]. On the other hand, anesthesia disrupts contextual effects like figure-ground modulation [26] and pattern motion [27], which are known to be mediated by top-down and recurrent connections. Other studies have shown that, at low concentrations, isoflurane anesthesia leaves the visual input to the cortex mostly intact, while the intracortical recurrent and top-down signals are disrupted [28, 29]. Thus, if the representation in the visual cortex is optimally sparse, disrupting the active sparsification by anesthesia should decrease sparseness.

We analyzed multi-unit neural activity from bundles of 16 electrodes implanted in primary visual cortex of 3 adult Long-Evans rats (5-11 units per recording session, for a total of 39 units). Recordings were made in the awake state and under four levels on anesthesia, from very light to deep (corresponding to concentrations of isoflurane between 0.6 and 2.0%) (see Suppl Mat for experimental details). In order to confirm that the effect of the anesthetic does not prevent visual information to reach the cortex, we presented the animals with a full-field periodic stimulus (flashing) at 3.75 Hz for 2 min in the awake state, and 3 min under anesthesia. The Fourier spectrum of the spikes trains on individual channels shows sharp peaks at the stimulation frequency in all states. We measured the response to the signal by the average amplitude of the Fourier spectrum between 3.7 and 3.8 Hz, and defined the amplitude of the noise, due to spontaneous activity and neural variability, as the average amplitude between 1 and 3.65 Hz (the amplitudes in this band are found to be noisy but uniform). The amplitude of the evoked signal decreases with increasing isoflurane concentration, due to a decrease in overall firing rate; however, the background noise is also suppressed with anesthesia, so that overall the signal-to-noise ratio does not decrease significantly with anesthesia (Fig. 6, ANOVA, P=0.46).

We recorded neural responses while the rats were shown a two minute movie recorded from a camera mounted on the head of a person walking in the woods. Neural activity was collected in 25 ms bins. All three sparseness measures increase significantly with increasing concentration of isoflurane[2] (Fig. 5B). Contrary to what is expected in a sparse-coding system, the data suggests that the contribution of lateral and top-down connections in the awake state leads to a less sparse code.

## 7 Discussion

We examined multi-electrode recordings from primary visual cortex of ferrets over development, and of rats at different levels of anesthesia. We found that, contrary to predictions based on theoretical considerations regarding optimal sparse coding systems, sparseness decreases with visual experience, and increases with increasing concentration of anesthetic. These data suggest that the

high sparseness levels that have been reported in previous accounts of sparseness in the visual cortex [7, 8, 9, 10, 11, 12], and which are otherwise consistent with our measurements (Fig. 3B, 5), are most likely a side effect of the high selectivity of neurons, or an overestimation due to the effect of anesthesia (Fig. 5; with the exception of [8], where sparseness was measured on awake animals), but do not indicate an active optimization of sparse responses (cf. [10]).

Our measurements of sparseness from neural data are based on multi-unit recording. By collecting spikes from multiple cells, we are in fact reporting a lower bound of the true sparseness values. While a precise measurement of the *absolute* value of these quantities would require single-unit measurement, our conclusions are based on *relative* comparisons of sparseness under different conditions, and are thus not affected.

Our theoretical predictions were verified with a common sparse coding model [3]. The model assumes linear summation in the generative process, and a particular sparse prior over the hidden unit. Despite these specific choices, we expect the model results to be general to the entire class of sparse coding models. In particular, the choice of comparing neural responses with Monte Carlo samples from the model's posterior distribution was taken in agreement with experimental results that report high neural variability. Alternatively, one could assume a deterministic neural architecture, with a network dynamic that would drive the activity of the units to values that maximize the image probability [3, 30, 31]. In this scenario, neural activity would converge to one of the modes of the distributions in Fig. 4, leading us to the same conclusions regarding the evolution of sparseness.

Although our analysis found no evidence for active sparsification in the primary visual cortex, ideas derived from and closely related to the sparse coding principle are likely to remain important for our understanding of visual processing. Efficient coding remains a most plausible functional account of coding in more peripheral parts of the sensory pathway, and particularly in the retina, from where raw visual input has to be sent through the bottleneck formed by the optic nerve without significant loss of information [32, 33]. Moreover, computational models of natural images are being extended from being strictly related to energy constraints and information transmission, to the more general view of density estimation in probabilistic, generative models [34, 35]. This view is compatible with our finding that the representation in the visual cortex becomes more dependent with age, and is less sparse in the awake condition than under anesthesia: We speculate that such dependencies reflect inference in a hierarchical generative model, where signals from lateral, recurrent connections in V1 and from feedback projections from higher areas are integrated with incoming evidence, in order to solve ambiguities at the level of basic image features using information from a global interpretation of the image [26, 19, 27, 20].

## Footnotes

[1]**Lifetime sparseness,** TR: effect of age is significant, Spearman's $\rho = -0.65$, $P < 0.01$; differences in mean between the four age groups in Fig. 3 are significant, ANOVA, $P = 0.02$, multiple comparison tests with Tukey-Kramer correction shows the mean of group P29-30 is different from that of groups P83-92 and P129-151 with $P < 0.05$; **Population sparseness,** TR: Spearman's $\rho = -0.75$, $P < 0.01$; ANOVA $P < 0.01$, multiple comparison shows the mean of group P29-30 is different from that of group P129-151 with $P < 0.05$; **Activity sparseness,** AS: Spearman's $\rho = -0.79$, $P < 0.01$; ANOVA $P < 0.01$, multiple comparison shows the mean of group P29-30 is different from that of groups P83-92 and P129-151 with $P < 0.05$.

[2]**Lifetime sparseness,** TR: ANOVA with different anesthesia groups, $P < 0.01$; multiple comparison tests with Tukey-Kramer correction shows the mean of awake group is different from the mean of all other groups with $P < 0.05$; **Population sparseness,** TR: ANOVA, $P < 0.01$; multiple comparison shows the mean of the awake group is different from that of the light, medium, and deep anesthesia groups, $P < 0.05$; **Activity sparseness,** AS: ANOVA $P < 0.01$, multiple comparison shows the mean of the awake group is different from that of the light, medium, and deep anesthesia groups, $P < 0.05$.

# References

[1] D.J. Field. What is the goal of sensory coding? *Neural Computation*, 6(4):559–601, 1994.

[2] B.A. Olshausen and D.J. Field. Sparse coding of sensory inputs. *Current Opinion in Neurobiology*, 14(4):481–487, 2004.

[3] B.A. Olshausen and D.J. Field. Emergence of simple-cell receptive field properties by learning a sparse code for natural images. *Nature*, 381(6583):607–609, 1996.

[4] A.J. Bell and T.J. Sejnowski. The 'independent components' of natural scenes are edge filters. *Vision Research*, 37(23):3327–3338, 1997.

[5] J.H. van Hateren and A. van der Schaaf. Independent component filters of natural images compared with simple cells in primary visual cortex. *Proc. R. Soc. Lond. B*, 265:359–366, 1998.

[6] A.S. Hsu and P. Dayan. An unsupervised learning model of neural plasticity: Orientation selectivity in goggle-reared kittens. *Vision Research*, 47(22):2868–2877, 2007.

[7] R. Baddeley, L.F. Abbott, M.C.A. Booth, F. Sengpiel, T. Freeman, E. Wakeman, and E.T. Rolls. Responses of neurons in primary and inferior temporal visual cortices to natural scenes. *Proceedings of the Royal Society B: Biological Sciences*, 264(1389):1775–1783, 1997.

[8] W.E. Vinje and J.L. Gallant. Sparse coding and decorrelation in primary visual cortex during natural vision. *Science*, 297(5456):1273–1276, 2000.

[9] M. Weliky, J. Fiser, R.H. Hunt, and D.N. Wagner. Coding of natural scenes in primary visual cortex. *Neuron*, 37(4):703–718, 2003.

[10] S.R. Lehky, T.J. Sejnowski, and R. Desimone. Selectivity and sparseness in the responses of striate complex cells. *Vision Research*, 45(1):57–73, 2005.

[11] S.C. Yen, J. Baker, and C.M. Gray. Heterogeneity in the responses of adjacent neurons to natural stimuli in cat striate cortex. *Journal of Neurophysiology*, 97(2):1326–1341, 2007.

[12] D.J. Tolhurst, D. Smyth, and I.D. Thompson. The sparseness of neuronal responses in ferret primary visual cortex. *Journal of Neuroscience*, 29(9):2355–2370, 2009.

[13] A. Treves and E.T. Rolls. What determines the capacity of autoassociative memories in the brain? *Network: Computation in Neural Systems*, 2(4):371–397, 1991.

[14] P. Foldiak and D. Endres. Sparse coding. *Scholarpedia*, 3(1):2984, 2008.

[15] B. Willmore and D.J. Tolhurst. Characterizing the sparseness of neural codes. *Network: Computation in Neural Systems*, 12:255–270, 2001.

[16] S. Osindero, M. Welling, and G.E. Hinton. Topographic product models applied to natural scene statistics. *Neural Computation*, 18:381–344, 2006.

[17] P. Berkes, R. Turner, and M. Sahani. On sparsity and overcompleteness in image models. In *Advances in Neural Information Processing Systems*, volume 20. MIT Press, 2008. Cambridge, MA.

[18] P.O. Hoyer and A. Hyvarinen. Interpreting neural response variability as monte carlo sampling of the posterior. In *Advances in Neural Information Processing Systems*, volume 15. MIT Press, 2003. Cambridge, MA.

[19] T.S. Lee and D. Mumford. Hierarchical Bayesian inference in the visual cortex. *Journal of the Optical Society of America A*, 20(7):1434–1448, 2003.

[20] P. Berkes, G. Orban, M. Lengyel, and J. Fiser. Matching spontaneous and evoked activity in V1: a hallmark of probabilistic inference. *Frontiers in Systems Neuroscience*, 2009. Conference Abstract: Computational and systems neuroscience.

[21] S. Kirkpatrick, C.D. Gelatt, and M.P. Vecchi. Optimization by simulated annealing. *Science*, 220:671–680, 1983.

[22] B. Chapman and M.P. Stryker. Development of orientation selectivity in ferret visual cortex and effects of deprivation. *Journal of Neuroscience*, 13:5251–5262, 1993.

[23] L.E. White, D.M. Coppola, and D. Fitzpatrick. The contribution of sensory experience to the maturation of orientation selectivity in ferret visual cortex. *Nature*, 411:1049–1052, 2001.

[24] P.H. Schiller, B.L. Finlay, and S.F. Volman. Quantitative studies of single-cell properties in monkey striate cortex. I. Spatiotemporal organization of receptive fields. *Journal of Neurophysiology*, 39(6):1288–1319, 1976.

[25] D.M. Snodderly and M. Gur. Organization of striate cortex of alert, trained monkeys (Macaca fascicularis): ongoing activity, stimulus selectivity, and widths of receptive field activating regions. *Journal of Neurophysiology*, 74(5):2100–2125, 1995.

[26] V.A.F. Lamme, K. Zipser, and H. Spekreijse. Figure-ground activity in primary visual cortex is suppressed by anesthesia. *PNAS*, 95:3263–3268, 1998.

[27] K. Guo, P.J. Benson, and C. Blakemore. Pattern motion is present in V1 of awake but not anaesthetized monkeys. *European Journal of Neuroscience*, 19:1055–1066, 2004.

[28] O. Detsch, C. Vahle-Hinz, E. Kochs, M. Siemers, and B. Bromm. Isoflurane induces dose-dependent changes of thalamic somatosensory information transfer. *Brain Research*, 829:77–89, 1999.

[29] H. Hentschke, C. Schwarz, and A. Bernd. Neocortex is the major target of sedative concentrations of volatile anaesthetics: strong depression of firing rates and increase of GABA-A receptor-mediated inhibition. *European Jounal of Neuroscience*, 21(1):93–102, 2005.

[30] P. Dayan and L.F. Abbott. *Theoretical Neuroscience: Computational and Mathematical Modeling of Neural Systems*. MIT Press, 2001.

[31] C.J. Rozell, D.H. Johnson, R.G. Baraniuk, and B.A. Olshausen. Sparse coding via thresholding and local competition in neural circuits. *Neural Computation*, 20:2526–2563, 2008.

[32] J.J. Atick. Could information theory provide an ecological theory of sensory processing? *Network: Computation in Neural Systems*, 3(2):213–251, 1992.

[33] V. Balasubramanian and M.J. Berry. Evidence for metabolically efficient codes in the retina. *Network: Computation in Neural Systems*, 13(4):531–553, 2002.

[34] Y. Karklin and M.S. Lewicki. A hierarchical bayesian model for learning non-linear statistical regularities in non-stationary natural signals. *Neural Computation*, 17(2):397–423, 2005.

[35] M.J. Wainwright and E.P. Simoncelli. Scale mixtures of gaussians and the statistics of natural images. In *Advances in Neural Information Processing Systems*. MIT Press, 2000. Cambridge, MA.

